# Learning Brain Connectivity of Alzheimer's Disease from Neuroimaging Data

**Shuai Huang[1], Jing Li[1], Liang Sun[2,3], Jun Liu[2,3], Teresa Wu[1], Kewei Chen[4], Adam Fleisher[4], Eric Reiman[4], Jieping Ye[2,3]**

[1]Industrial Engineering, [2]Computer Science and Engineering, and [3]Center for Evolutionary Functional Genomics, The Biodesign Institute, Arizona State University, Tempe, USA

{shuang31, jing.li.8, sun.liang, j.liu, teresa.wu, jieping.ye}@asu.edu

[4]Banner Alzheimer's Institute and Banner PET Center, Banner Good Samaritan Medical Center, Phoenix, USA

{kewei.chen, adam.fleisher, eric.reiman}@bannerhealth.com

## Abstract

Recent advances in neuroimaging techniques provide great potentials for effective diagnosis of Alzheimer's disease (AD), the most common form of dementia. Previous studies have shown that AD is closely related to the alternation in the functional brain network, i.e., the functional connectivity among different brain regions. In this paper, we consider the problem of learning functional brain connectivity from neuroimaging, which holds great promise for identifying image-based markers used to distinguish Normal Controls (NC), patients with Mild Cognitive Impairment (MCI), and patients with AD. More specifically, we study sparse inverse covariance estimation (SICE), also known as exploratory Gaussian graphical models, for brain connectivity modeling. In particular, we apply SICE to learn and analyze functional brain connectivity patterns from different subject groups, based on a key property of SICE, called the "monotone property" we established in this paper. Our experimental results on neuroimaging PET data of 42 AD, 116 MCI, and 67 NC subjects reveal several interesting connectivity patterns consistent with literature findings, and also some new patterns that can help the knowledge discovery of AD.

## 1 Introduction

Alzheimer's disease (AD) is a fatal, neurodegenerative disorder characterized by progressive impairment of memory and other cognitive functions. It is the most common form of dementia and currently affects over five million Americans; this number will grow to as many as 14 million by year 2050. The current knowledge about the cause of AD is very limited; clinical diagnosis is imprecise with definite diagnosis only possible by autopsy; also, there is currently no cure for AD, while most drugs only alleviate the symptoms.

To tackle these challenging issues, the rapidly advancing neuroimaging techniques provide great potentials. These techniques, such as MRI, PET, and fMRI, produce data (images) of brain structure and function, making it possible to identify the difference between AD and normal brains. Recent studies have demonstrated that neuroimaging data provide more sensitive and consistent measures of AD onset and progression than conventional clinical

assessment and neuropsychological tests [1].

Recent studies have found that AD is closely related to the alternation in the functional brain network, i.e., the functional connectivity among different brain regions [2]-[3]. Specifically, it has been shown that functional connectivity substantially decreases between the hippocampus and other regions of AD brains [3]-[4]. Also, some studies have found increased connectivity between the regions in the frontal lobe [6]-[7].

Learning functional brain connectivity from neuroimaging data holds great promise for identifying image-based markers used to distinguish among AD, MCI (Mild Cognitive Impairment), and normal aging. Note that MCI is a transition stage from normal aging to AD. Understanding and precise diagnosis of MCI have significant clinical value since it can serve as an early warning sign of AD. Despite all these, existing research in functional brain connectivity modeling suffers from limitations. A large body of functional connectivity modeling has been based on correlation analysis [2]-[3], [5]. However, correlation only captures pairwise information and fails to provide a complete account for the interaction of many (more than two) brain regions. Other multivariate statistical methods have also been used, such as Principle Component Analysis (PCA) [8], PCA-based Scaled Subprofile Model [9], Independent Component Analysis [10]-[11], and Partial Least Squares [12]-[13], which group brain regions into latent components. The brain regions within each component are believed to have strong connectivity, while the connectivity between components is weak. One major drawback of these methods is that the latent components may not correspond to any biological entities, causing difficulty in interpretation. In addition, graphical models have been used to study brain connectivity, such as structural equation models [14]-[15], dynamic causal models [16], and Granger causality. However, most of these approaches are confirmative, rather than exploratory, in the sense that they require a prior model of brain connectivity to begin with. This makes them inadequate for studying AD brain connectivity, because there is little prior knowledge about which regions should be involved and how they are connected. This makes exploratory models highly desirable.

In this paper, we study sparse inverse covariance estimation (SICE), also known as exploratory Gaussian graphical models, for brain connectivity modeling. Inverse covariance matrix has a clear interpretation that the off-diagonal elements correspond to partial correlations, i.e., the correlation between each pair of brain regions given all other regions. This provides a much better model for brain connectivity than simple correlation analysis which models each pair of regions without considering other regions. Also, imposing sparsity on the inverse covariance estimation ensures a reliable brain connectivity to be modeled with limited sample size, which is usually the case in AD studies since clinical samples are difficult to obtain. From a domain perspective, imposing sparsity is also valid because neurological findings have demonstrated that a brain region usually only directly interacts with a few other brain regions in neurological processes [2]-[3]. Various algorithms for achieving SICE have been developed in recent year [17]-[22]. In addition, SICE has been used in various applications [17], [21], [23]-[26].

In this paper, we apply SICE to learn functional brain connectivity from neuroimaging and analyze the difference among AD, MCI, and NC based on a key property of SICE, called the "monotone property" we established in this paper. Unlike the previous study which is based on a specific level of sparsity [26], the monotone property allows us to study the connectivity pattern using different levels of sparsity and obtain an order for the strength of connection between pairs of brain regions. In addition, we apply bootstrap hypothesis testing to assess the significance of the connection. Our experimental results on PET data of 42 AD, 116 MCI, and 67 NC subjects enrolled in the Alzheimer's Disease Neuroimaging Initiative project reveal several interesting connectivity patterns consistent with literature findings, and also some new patterns that can help the knowledge discovery of AD.

## 2    SICE: Background and the Monotone Property

An inverse covariance matrix can be represented graphically. If used to represent brain connectivity, the nodes are activated brain regions; existence of an arc between two nodes means that the two brain regions are closely related in the brain's functional process.

Let $\{X_1, \cdots, X_p\}$ be all the brain regions under study. We assume that $\{X_1, \cdots, X_p\}$ follows a multivariate Gaussian distribution with mean $\boldsymbol{\mu}$ and covariance matrix $\boldsymbol{\Sigma}$. Let $\boldsymbol{\Theta} = \boldsymbol{\Sigma}^{-1}$ be the inverse covariance matrix. Suppose we have $n$ samples (e.g., $n$ subjects with AD) for these brain regions. Note that we will only illustrate here the SICE for AD, whereas the SICE for MCI and NC can be achieved in a similar way.

We can formulate the SICE into an optimization problem, i.e.,

$$\widehat{\boldsymbol{\Theta}} = \underset{\boldsymbol{\Theta} > 0}{\text{argmax}} \; log\big(det(\boldsymbol{\Theta})\big) - tr(\mathbf{S}\boldsymbol{\Theta}) - \lambda \|vec(\boldsymbol{\Theta})\|_1 \tag{1}$$

where $\mathbf{S}$ is the sample covariance matrix; $det(\cdot)$, $tr(\cdot)$, and $\|vec(\cdot)\|_1$ denote the determinant, trace, and sum of the absolute values of all elements of a matrix, respectively. The part "$log\big(det(\boldsymbol{\Theta})\big) - tr(\mathbf{S}\boldsymbol{\Theta})$" in (1) is the log-likelihood, whereas the part "$\|vec(\boldsymbol{\Theta})\|_1$" represents the "sparsity" of the inverse covariance matrix $\boldsymbol{\Theta}$. (1) aims to achieve a tradeoff between the likelihood fit of the inverse covariance estimate and the sparsity. The tradeoff is controlled by $\lambda$, called the regularization parameter; larger $\lambda$ will result in more sparse estimate for $\boldsymbol{\Theta}$. The formulation in (1) follows the same line of the $L_1$-norm regularization, which has been introduced into the least squares formulation to achieve model sparsity and the resulting model is called Lasso [27]. We employ the algorithm in [19] in this paper. Next, we show that with $\lambda$ going from small to large, the resulting brain connectivity models have a monotone property. Before introducing the monotone property, the following definitions are needed.

**Definition**: In the graphical representation of the inverse covariance, if node $X_i$ is connected to $X_j$ by an arc, then $X_i$ is called a "neighbor" of $X_j$. If $X_i$ is connected to $X_k$ though some chain of arcs, then $X_i$ is called a "connectivity component" of $X_k$.

Intuitively, being neighbors means that two nodes (i.e., brain regions) are directly connected, whereas being connectivity components means that two brain regions are indirectly connected, i.e., the connection is mediated through other regions. In other words, not being connectivity components (i.e., two nodes completely separated in the graph) means that the two corresponding brain regions are completely independent of each other. Connectivity components have the following monotone property:

**Monotone property of SICE**: Let $\mathbf{C}_k(\lambda_1)$ and $\mathbf{C}_k(\lambda_2)$ be the sets of all the connectivity components of $X_k$ with $\lambda = \lambda_1$ and $\lambda = \lambda_2$, respectively. If $\lambda_1 < \lambda_2$, then $\mathbf{C}_k(\lambda_1) \supseteq \mathbf{C}_k(\lambda_2)$.

Intuitively, if two regions are connected (either directly or indirectly) at one level of sparseness ($\lambda = \lambda_2$), they will be connected at all lower levels of sparseness ($\lambda < \lambda_2$). Proof of the monotone property can be found in the supplementary file [29]. This monotone property can be used to identify how strongly connected each node (brain region) $X_k$ to its connectivity components. For example, assuming that $\mathbf{C}_k(\lambda_1) = \{X_i, X_j\}$ and $\mathbf{C}_k(\lambda_2) = \{X_i\}$, this means that $X_i$ is more strongly connected to $X_k$ than $X_j$. Thus, by changing $\lambda$ from small to large, we can obtain an order for the strength of connection between pairs of brain regions. As will be shown in Section 3, this order is different among AD, MCI, and NC.

# 3 Application in Brain Connectivity Modeling of AD

## 3.1 Data acquisition and preprocessing

We apply SICE on FDG-PET images for 49 AD, 116 MCI, and 67 NC subjects downloaded from the ADNI website. We apply Automated Anatomical Labeling (AAL) [28] to extract data from each of the 116 anatomical volumes of interest (AVOI), and derived average of each AVOI for every subject. The AVOIs represent different regions of the whole brain.

## 3.2 Brain connectivity modeling by SICE

42 AVOIs are selected for brain connectivity modeling, as they are considered to be potentially related to AD. These regions distribute in the frontal, parietal, occipital, and temporal lobes. Table 1 list of the names of the AVOIs with their corresponding lobes. The number before each AVOI is used to index the node in the connectivity models.

We apply the SICE algorithm to learn one connectivity model for AD, one for MCI, and one for NC, for a given $\lambda$. With different $\lambda$'s, the resulting connectivity models hold a monotone property, which can help obtain an order for the strength of connection between brain regions. To show the order clearly, we develop a tree-like plot in Fig. 1, which is for the AD group. To generate this plot, we start $\lambda$ at a very small value (i.e., the right-most of the horizontal axis), which results in a fully-connected connectivity model. A fully-connected connectivity model is one that contains no region disconnected with the rest of the brain. Then, we decrease $\lambda$ by small steps and record the order of the regions disconnected with the rest of the brain regions.

Table 1: Names of the AVOIs for connectivity modeling ("L" means that the brain region is located at the left hemisphere; "R" means right hemisphere.)

| Frontal lobe | | Parietal lobe | | Occipital lobe | | Temporal lobe | |
|---|---|---|---|---|---|---|---|
| 1 | Frontal_Sup_L | 13 | Parietal_Sup_L | 21 | Occipital_Sup_L | 27 | Temporal_Sup_L |
| 2 | Frontal_Sup_R | 14 | Parietal_Sup_R | 22 | Occipital_Sup_R | 28 | Temporal_Sup_R |
| 3 | Frontal_Mid_L | 15 | Parietal_Inf_L | 23 | Occipital_Mid_L | 29 | Temporal_Pole_Sup_L |
| 4 | Frontal_Mid_R | 16 | Parietal_Inf_R | 24 | Occipital_Mid_R | 30 | Temporal_Pole_Sup_R |
| 5 | Frontal_Sup_Medial_L | 17 | Precuneus_L | 25 | Occipital_Inf_L | 31 | Temporal_Mid_L |
| 6 | Frontal_Sup_Medial_R | 18 | Precuneus_R | 26 | Occipital_Inf_R | 32 | Temporal_Mid_R |
| 7 | Frontal_Mid_Orb_L | 19 | Cingulum_Post_L | | | 33 | Temporal_Pole_Mid_L |
| 8 | Frontal_Mid_Orb_R | 20 | Cingulum_Post_R | | | 34 | Temporal_Pole_Mid_R |
| 9 | Rectus_L | | | | | 35 | Temporal_Inf_L 8301 |
| 10 | Rectus_R | | | | | 36 | Temporal_Inf_R 8302 |
| 11 | Cingulum_Ant_L | | | | | 37 | Fusiform_L |
| 12 | Cingulum_Ant_R | | | | | 38 | Fusiform_R |
| | | | | | | 39 | Hippocampus_L |
| | | | | | | 40 | Hippocampus_R |
| | | | | | | 41 | ParaHippocampal_L |
| | | | | | | 42 | ParaHippocampal_R |

For example, in Fig. 1, as $\lambda$ decreases below $\lambda_1$ (but still above $\lambda_2$), region "Tempora_Sup_L" is the first one becoming disconnected from the rest of the brain. As $\lambda$ decreases below $\lambda_2$ (but still above $\lambda_3$), the rest of the brain further divides into three disconnected clusters, including the cluster of "Cingulum_Post_R" and "Cingulum_Post_L", the cluster of "Fusiform_R" up to "Hippocampus_L", and the cluster of the other regions. As $\lambda$ continuously decreases, each current cluster will split into smaller clusters; eventually, when $\lambda$ reaches a very large value, there will be no arc in the IC model, i.e., each region is now a cluster of itself and the split will stop. The sequence of the splitting gives an order for the strength of connection between brain regions. Specifically, the earlier (i.e., smaller $\lambda$) a region or a cluster of regions becomes disconnected from the rest of the brain, the weaker it is connected with the rest of the brain. For example, in Fig. 1, it can be known that "Tempora_Sup_L" may be the weakest region in the brain network of AD; the second weakest ones are the cluster of "Cingulum_Post_R" and "Cingulum_Post_L", and the cluster of "Fusiform_R" up to "Hippocampus_L". It is very interesting to see that the weakest and second weakest brain regions in the brain network include "Cingulum_Post_R" and "Cingulum_Post_L" as well as regions all in the temporal lobe, all of which have been found to be affected by AD early and severely [3]-[5].

Next, to facilitate the comparison between AD and NC, a tree-like plot is also constructed for NC, as shown in Fig. 2. By comparing the plots for AD and NC, we can observe the following two distinct phenomena: First, in AD, between-lobe connectivity tends to be weaker than within-lobe connectivity. This can be seen from Fig. 1 which shows a clear pattern that the lobes become disconnected with each other before the regions within each lobe become disconnected with each other, as $\lambda$ goes from small to large. This pattern does not show in Fig. 2 for NC. Second, the same brain regions in the left and right hemisphere are connected much weaker in AD than in NC. This can be seen from Fig. 2 for NC, in which the same brain regions in the left and right hemisphere are still connected even at a very large $\lambda$ for NC. However, this pattern does not show in Fig. 1 for AD.

Furthermore, a tree-like plot is also constructed for MCI (Fig. 3), and compared with the plots for AD and NC. In terms of the two phenomena discussed previously, MCI shows similar patterns to AD, but these patterns are not as distinct from NC as AD. Specifically, in terms of the first

phenomenon, MCI also shows weaker between-lobe connectivity than within-lobe connectivity, which is similar to AD. However, the degree of weakerness is not as distinctive as AD. For example, a few regions in the temporal lobe of MCI, including "Temporal_Mid_R" and "Temporal_Sup_R", appear to be more strongly connected with the occipital lobe than with other regions in the temporal lobe. In terms of the second phenomenon, MCI also shows weaker between-hemisphere connectivity in the same brain region than NC. However, the degree of weakerness is not as distinctive as AD. For example, several left-right pairs of the same brain regions are still connected even at a very large $\lambda$, such as "Rectus_R" and "Rectus_L", "Frontal_Mid_Orb_R" and "Frontal_Mid_Orb _L", "Parietal_Sup_R" and "Parietal_Sup_L", as well as "Precuneus_R" and "Precuneus_L". All above findings are consistent with the knowledge that MCI is a transition stage between normal aging and AD.

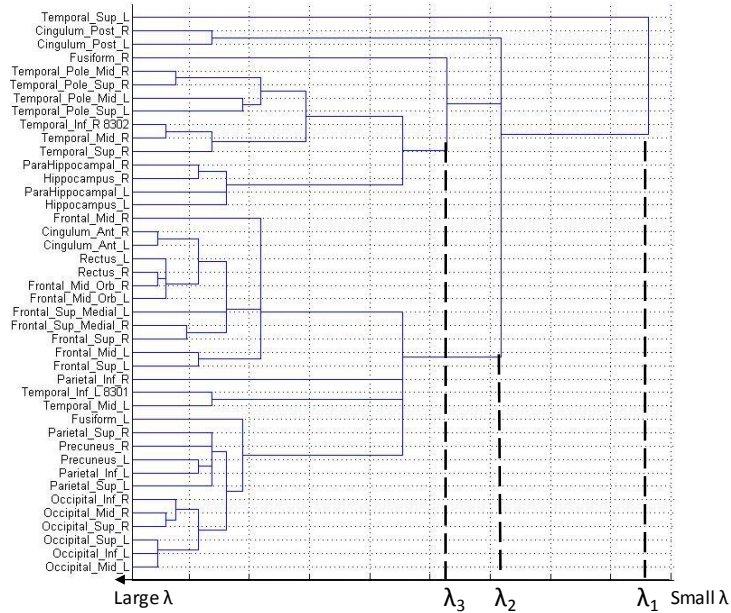

Fig 1: Order for the strength of connection between brain regions of AD

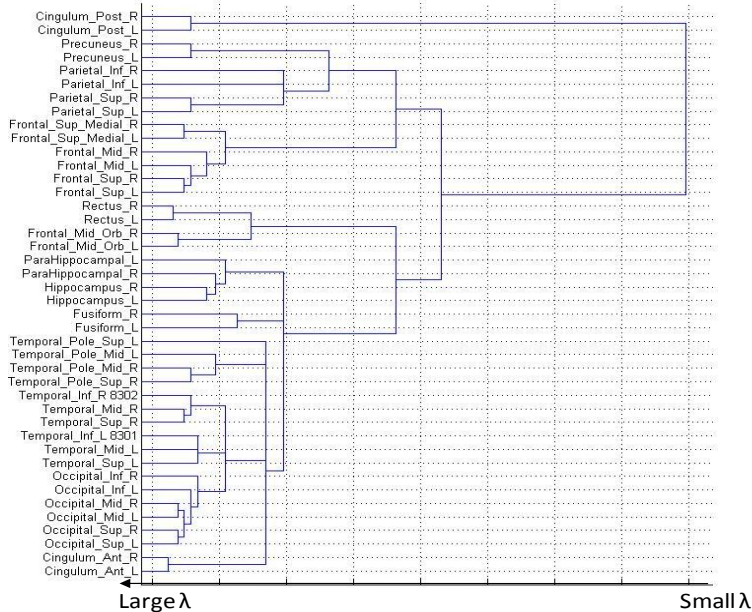

Fig 2: Order for the strength of connection between brain regions of NC

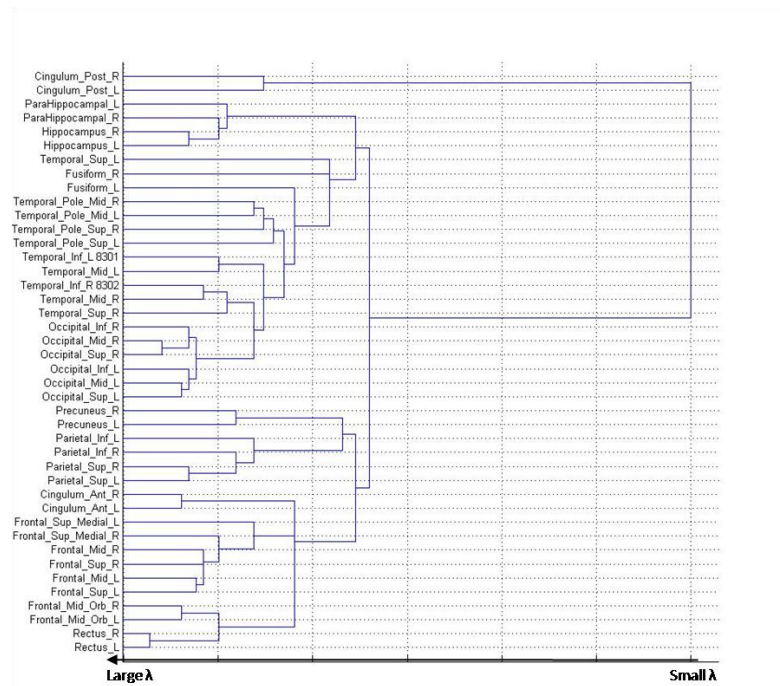

Fig 3: Order for the strength of connection between brain regions of MCI

Furthermore, we would like to compare how within-lobe and between-lobe connectivity is different across AD, MCI, and NC. To achieve this, we first learn one connectivity model for AD, one for MCI, and one for NC. We adjust the $\lambda$ in the learning of each model such that the three models, corresponding to AD, MCI, and NC, respectively, will have the same total number of arcs. This is to "normalize" the models, so that the comparison will be more focused on how the arcs distribute differently across different models. By selecting different values for the total number of arcs, we can obtain models representing the brain connectivity at different levels of strength. Specifically, given a small value for the total number of arcs, only strong arcs will show up in the resulting connectivity model, so the model is a model of strong brain connectivity; when increasing the total number of arcs, mild arcs will also show up in the resulting connectivity model, so the model is a model of mild and strong brain connectivity.

For example, Fig. 4 shows the connectivity models for AD, MCI, and NC with the total number of arcs equal to 50 (Fig. 4(a)), 120 (Fig. 4(b)), and 180 (Fig. 4(c)). In this paper, we use a "matrix" representation for the SICE of a connectivity model. In the matrix, each row represents one node and each column also represents one node. Please see Table 1 for the correspondence between the numbering of the nodes and the brain region each number represents. The matrix contains black and white cells: a black cell at the $i$-th row, $j$-th column of the matrix represents existence of an arc between nodes $X_i$ and $X_j$ in the SICE-based connectivity model, whereas a white cell represents absence of an arc. According to this definition, the total number of black cells in the matrix is equal to twice the total number of arcs in the SICE-based connectivity model. Moreover, on each matrix, four red cubes are used to highlight the brain regions in each of the four lobes; that is, from top-left to bottom-right, the red cubes highlight the frontal, parietal, occipital, and temporal lobes, respectively. The black cells inside each red cube reflect within-lobe connectivity, whereas the black cells outside the cubes reflect between-lobe connectivity.

While the connectivity models in Fig. 4 clearly show some connectivity difference between AD, MCI, and NC, it is highly desirable to test if the observed difference is statistically significant. Therefore, we further perform a hypothesis testing and the results are summarized in Table 2. Specifically, a P-value is recorded in the sub-table if it is smaller than 0.1, such a P-value is further highlighted if it is even smaller than 0.05; a "---" indicates that the corresponding test is not significant (P-value>0.1). We can observe from Fig. 4 and Table 2:

*Within-lobe connectivity*: The temporal lobe of AD has significantly less connectivity than NC. This is true across different strength levels (e.g., strong, mild, and weak) of the connectivity; in other words, even the connectivity between some strongly-connected brain regions in the temporal lobe may be disrupted by AD. In particular, it is clearly from Fig. 4(b) that the regions "Hippocampus" and "ParaHippocampal" (numbered by 39-42, located at the right-bottom corner of Fig. 4(b)) are much more separated from other regions in AD than in NC. The decrease in connectivity in the temporal lobe of AD, especially between the Hippocampus and other regions, has been extensively reported in the literature [3]-[5]. Furthermore, the temporal lobe of MCI does not show a significant decrease in connectivity, compared with NC. This may be because MCI does not disrupt the temporal lobe as badly as AD.

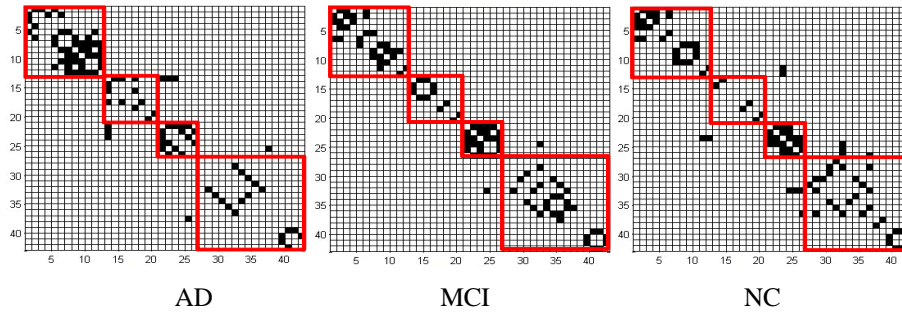

Fig 4(a): SICE-based brain connectivity models (total number of arcs equal to 50)

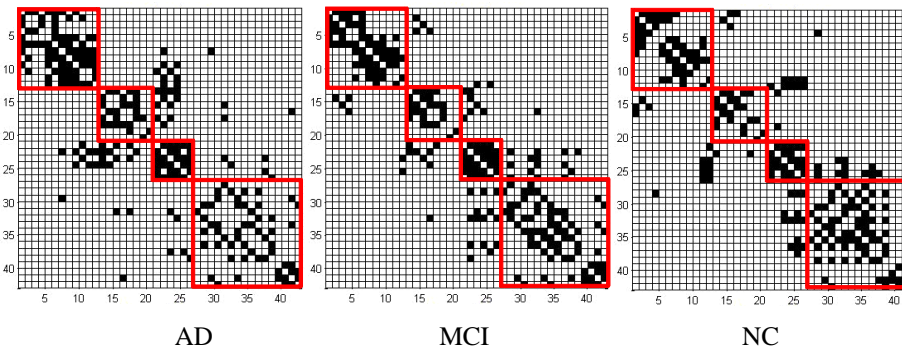

Fig 4(b): SICE-based brain connectivity models (total number of arcs equal to 120)

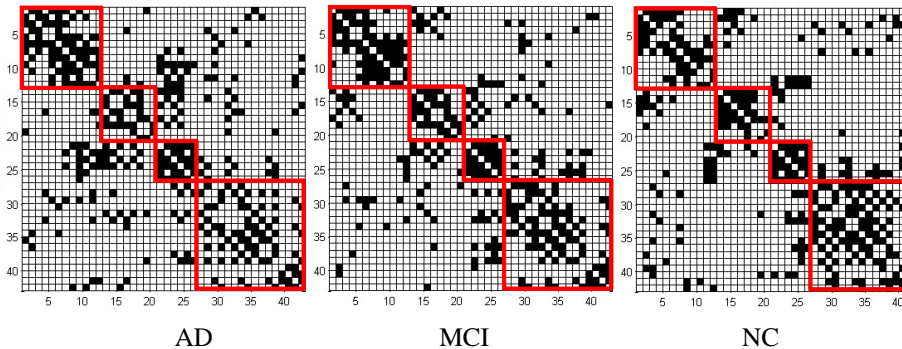

Fig 4(c): SICE-based brain connectivity models (total number of arcs equal to 180)

The frontal lobe of AD has significantly more connectivity than NC, which is true across different strength levels of the connectivity. This has been interpreted as compensatory reallocation or recruitment of cognitive resources [6]-[7]. Because the regions in the frontal lobe are typically affected later in the course of AD (our data are early AD), the increased connectivity in the frontal lobe may help preserve some cognitive functions in AD patients. Furthermore, the frontal lobe of MCI does not show a significant increase in connectivity, compared with NC. This indicates that the compensatory effect in MCI brain may not be as strong as that in AD brains.

Table 2: P-values from the statistical significance test of connectivity difference among AD, MCI, and NC

(a) Total number of arcs = 50

| AD vs. NC | Frontal | Parietal | Occipital | Temporal |
|---|---|---|---|---|
| Frontal | 0.053 | --- | --- | --- |
| Parietal | | --- | 0.083 | --- |
| Occipital | | | --- | --- |
| Temporal | | | | 0.091 |

| AD vs. MCI | Frontal | Parietal | Occipital | Temporal |
|---|---|---|---|---|
| Frontal | 0.085 | --- | --- | --- |
| Parietal | | --- | 0.088 | --- |
| Occipital | | | 0.054 | --- |
| Temporal | | | | 0.093 |

| MCI vs. NC | Frontal | Parietal | Occipital | Temporal |
|---|---|---|---|---|
| Frontal | --- | --- | --- | --- |
| Parietal | | --- | --- | --- |
| Occipital | | | --- | --- |
| Temporal | | | | --- |

(b) Total number of arcs = 120

| AD vs. NC | Frontal | Parietal | Occipital | Temporal |
|---|---|---|---|---|
| Frontal | 0.033 | --- | --- | --- |
| Parietal | | --- | 0.007 | 0.095 |
| Occipital | | | --- | 0.018 |
| Temporal | | | | 0.039 |

| AD vs. MCI | Frontal | Parietal | Occipital | Temporal |
|---|---|---|---|---|
| Frontal | --- | 0.055 | 0.022 | --- |
| Parietal | | --- | --- | --- |
| Occipital | | | --- | 0.019 |
| Temporal | | | | --- |

| MCI vs. NC | Frontal | Parietal | Occipital | Temporal |
|---|---|---|---|---|
| Frontal | --- | --- | 0.024 | --- |
| Parietal | | --- | 0.052 | --- |
| Occipital | | | --- | --- |
| Temporal | | | | --- |

(c) Total number of arcs = 180

| AD vs. NC | Frontal | Parietal | Occipital | Temporal |
|---|---|---|---|---|
| Frontal | 0.067 | 0.058 | 0.096 | --- |
| Parietal | | --- | 0.019 | 0.011 |
| Occipital | | | --- | --- |
| Temporal | | | | 0.017 |

| AD vs. MCI | Frontal | Parietal | Occipital | Temporal |
|---|---|---|---|---|
| Frontal | --- | 0.063 | 0.004 | --- |
| Parietal | | --- | --- | --- |
| Occipital | | | --- | 0.058 |
| Temporal | | | | --- |

| MCI vs. NC | Frontal | Parietal | Occipital | Temporal |
|---|---|---|---|---|
| Frontal | --- | --- | 0.061 | --- |
| Parietal | | --- | 0.0504 | --- |
| Occipital | | | --- | 0.041 |
| Temporal | | | | --- |

There is no significant difference among AD, MCI, and NC in terms of the connectivity within the parietal lobe and within the occipital lobe. Another interesting finding is that all the P-values in the third sub-table of Table 2(a) are insignificant. This implies that distribution of the strong connectivity within and between lobes for MCI is very similar to NC; in other words, MCI has not been able to disrupt the strong connectivity among brain regions (it disrupts some mild and weak connectivity though).

_Between-lobe connectivity_: In general, human brains tend to have less between-lobe connectivity than within-lobe connectivity. A majority of the strong connectivity occurs within lobes, but rarely between lobes. These can be clearly seen from Fig. 4 (especially Fig. 4(a)) in which there are much more black cells along the diagonal direction than the off-diagonal direction, regardless of AD, MCI, and NC.

The connectivity between the parietal and occipital lobes of AD is significantly more than NC which is true especially for mild and weak connectivity. The increased connectivity between the parietal and occipital lobes of AD has been previously reported in [3]. It is also interpreted as a compensatory effect in [6]-[7]. Furthermore, MCI also shows increased connectivity between the parietal and occipital lobes, compared with NC, but the increase is not as significant as AD.

While the connectivity between the frontal and occipital lobes shows little difference between AD and NC, such connectivity for MCI shows a significant decrease especially for mild and weak connectivity. Also, AD may have less temporal-occipital connectivity, less frontal-parietal connectivity, but more parietal-temporal connectivity than NC.

_Between-hemisphere connectivity_: Recall that we have observed from the tree-like plots in Figs. 3 and 4 that the same brain regions in the left and right hemisphere are connected much weaker in AD than in NC. It is desirable to test if this observed difference is statistically significant. To achieve this, we test the statistical significance of the difference among AD, MCI, and NC, in term of the number of connected same-region left-right pairs. Results show that when the total number of arcs in the connectivity models is equal to 120 or 90, none of the tests is significant. However, when the total number of arcs is equal to 50, the P-values of the tests for "AD vs. NC", "AD vs. MCI", and "MCI vs. NC" are 0.009, 0.004, and 0.315, respectively. We further perform tests for the total number of arcs equal to 30 and find the P-values to be 0. 0055, 0.053, and 0.158, respectively. These results indicate that AD disrupts the strong connectivity between the same regions of the left and right hemispheres, whereas this disruption is not significant in MCI.

# 4      Conclusion

In the paper, we applied SICE to model functional brain connectivity of AD, MCI, and NC based on PET neuroimaging data, and analyze the patterns based on the monotone property of SICE. Our findings were consistent with the previous literature and also showed some new aspects that may suggest further investigation in brain connectivity research in the future.

# References

[1] S. Molchan. (2005) The Alzheimer's disease neuroimaging initiative. Business Briefing: US Neurology Review, pp.30-32, 2005.

[2] C.J. Stam, B.F. Jones, G. Nolte, M. Breakspear, and P. Scheltens. (2007) Small-world networks and functional connectivity in Alzheimer's disease. Cerebral Corter 17:92-99.

[3] K. Supekar, V. Menon, D. Rubin, M. Musen, M.D. Greicius. (2008)    Network Analysis of Intrinsic Functional Brain Connectivity in Alzheimer's Disease. PLoS Comput Biol 4(6) 1-11.

[4] K. Wang, M. Liang, L. Wang, L. Tian, X. Zhang, K. Li and T. Jiang. (2007) Altered Functional Connectivity in Early Alzheimer's Disease: A Resting-State fMRI Study, Human Brain Mapping 28, 967-978.

[5] N.P. Azari, S.I. Rapoport, C.L. Grady, M.B. Schapiro, J.A. Salerno, A. Gonzales-Aviles. (1992) Patterns of interregional correlations of cerebral glucose metabolic rates in patients with dementia of the Alzheimer type. Neurodegeneration 1: 101–111.

[6] R.L. Gould, B.Arroyo, R,G. Brown, A.M. Owen, E.T. Bullmore and R.J. Howard. (2006) Brain Mechanisms of Successful Compensation during Learning in Alzheimer Disease, Neurology 67, 1011-1017.

[7] Y. Stern. (2006) Cognitive Reserve and Alzheimer Disease, Alzheimer Disease Associated Disorder 20, 69-74.

[8] K.J. Friston. (1994) Functional and effective connectivity: A synthesis. Human Brain Mapping 2, 56-78.

[9] G. Alexander, J. Moeller. (1994) Application of the Scaled Subprofile model: a statistical approach to the analysis of functional patterns in neuropsychiatric disorders: A principal component approach to modeling regional patterns of brain function in disease. Human Brain Mapping, 79-94.

[10] V.D. Calhoun, T. Adali, G.D. Pearlson, J.J. Pekar. (2001) Spatial and temporal independent component analysis of functional MRI data containing a pair of task-related waveforms. Hum.Brain Mapp. 13, 43-53.

[11] V.D. Calhoun, T. Adali, J.J. Pekar, G.D. Pearlson. (2003) Latency (in)sensitive ICA. Group independent component analysis of fMRI data in the temporal frequency domain. Neuroimage. 20, 1661-1669.

[12] A.R. McIntosh, F.L. Bookstein, J.V. Haxby, C.L. Grady. (1996) Spatial pattern analysis of functional brain images using partial least squares. Neuroimage. 3, 143-157.

[13] K.J. Worsley, J.B. Poline, K.J. Friston, A.C. Evans. (1997) Characterizing the response of PET and fMRI data using multivariate linear models. Neuroimage. 6, 305-319.

[14] E. Bullmore, B. Horwitz, G. Honey, M. Brammer, S. Williams, T. Sharma. (2000) How good is good enough in path analysis of fMRI data? NeuroImage 11, 289–301.

[15] A.R. McIntosh, C.L. Grady, L.G. Ungerieider, J.V. Haxby, S.I. Rapoport, B. Horwitz. (1994) Network analysis of cortical visual pathways mapped with PET. J. Neurosci. 14 (2), 655–666.

[16] K.J. Friston, L. Harrison, W. Penny. (2003) Dynamic causal modelling. Neuroimage 19, 1273-1302.

[17] O. Banerjee, L. El Ghaoui, and A. d'Aspremont. (2008) Model selection through sparse maximum likelihood estimation for multivariate gaussian or binary data. Journal of Machine Learning Research 9:485-516.

[18] J. Dahl, L. Vandenberghe, and V. Roycowdhury. (2008) Covariance selection for nonchordal graphs via chordal embedding. Optimization Methods Software 23(4):501-520.

[19] J. Friedman, T.astie, and R. Tibsirani. (2007) Spares inverse covariance estimation with the graphical lasso, Biostatistics 8(1):1-10.

[20] J.Z. Huang, N. Liu, M. Pourahmadi, and L. Liu. (2006) Covariance matrix selection and estimation via penalized normal likelihood. Biometrika, 93(1):85-98.

[21] H. Li and J. Gui. (2005) Gradient directed regularization for sparse Gaussian concentration graphs, with applications to inference of genetic networks. Biostatistics 7(2):302-317.

[22] Y. Lin. (2007) Model selection and estimation in the gaussian graphical model. Biometrika 94(1)19-35, 2007.

[23] A. Dobra, C. Hans, B. Jones, J.R. Nevins, G. Yao, and M. West. (2004) Sparse graphical models for exploring gene expression data. Journal of Multivariate Analysis 90(1):196-212.

[24] A. Berge, A.C. Jensen, and A.H.S. Solberg. (2007) Sparse inverse covariance estimates for hyperspectral image classification, Geoscience and Remote Sensing, IEEE Transactions on, 45(5):1399-1407.

[25] J.A. Bilmes. (2000) Factored sparse inverse covariance matrices. In ICASSP:1009-1012.

[26] L. Sun and et al. (2009) Mining Brain Region Connectivity for Alzheimer's Disease Study via Sparse Inverse Covariance Estimation. In KDD: 1335-1344.

[27] R. Tibshirani. (1996) Regression shrinkage and selection via the lasso. Journal of the Royal Statistical Society Series B 58(1):267-288.

[28] N. Tzourio-Mazoyer and et al. (2002) Automated anatomical labeling of activations in SPM using a macroscopic anatomical parcellation of the MNI MRI single subject brain. Neuroimage 15:273-289.

[29] Supplemental information for "Learning Brain Connectivity of Alzheimer's Disease from Neuroimaging Data". http://www.public.asu.edu/~jye02/Publications/AD-supplemental-NIPS09.pdf
